# Fault Diagnosis of Antenna Pointing Systems using Hybrid Neural Network and Signal Processing Models

**Padhraic Smyth, Jeff Mellstrom**
Jet Propulsion Laboratory 238-420
California Institute of Technology
Pasadena, CA 91109

## Abstract

We describe in this paper a novel application of neural networks to system health monitoring of a large antenna for deep space communications. The paper outlines our approach to building a monitoring system using hybrid signal processing and neural network techniques, including autoregressive modelling, pattern recognition, and Hidden Markov models. We discuss several problems which are somewhat generic in applications of this kind — in particular we address the problem of detecting classes which were not present in the training data. Experimental results indicate that the proposed system is sufficiently reliable for practical implementation.

## 1    Background: The Deep Space Network

The Deep Space Network (DSN) (designed and operated by the Jet Propulsion Laboratory (JPL) for the National Aeronautics and Space Administration (NASA)) is unique in terms of providing end-to-end telecommunication capabilities between earth and various interplanetary spacecraft throughout the solar system. The ground component of the DSN consists of three ground station complexes located in California, Spain and Australia, giving full 24-hour coverage for deep space communications. Since spacecraft are always severely limited in terms of available transmitter power (for example, each of the Voyager spacecraft only use 20 watts to transmit signals back to earth), all subsystems of the end-to-end communications link (radio telemetry, coding, receivers, amplifiers) tend to be pushed to the absolute limits of performance. The large steerable ground antennas (70m and 34m dishes) represent critical potential single points of failure in the network. In particular there is only a single 70m antenna at each complex because of the large cost and calibration effort involved in constructing and operating a steerable antenna of that size — the entire structure (including pedestal support) weighs over 8,000 tons.

The antenna pointing systems consist of azimuth and elevation axes drives which respond to computer-generated trajectory commands to steer the antenna in real-time. Pointing accuracy requirements for the antenna are such that there is little tolerance for component degradation. Achieving the necessary degree of positional accuracy is rendered difficult by various non-linearities in the gear and motor elements and environmental disturbances such as gusts of wind affecting the antenna dish structure. Off-beam pointing can result in rapid fall-off in signal-to-noise ratios and consequent potential loss of irrecoverable scientific data from the spacecraft.

The pointing systems are a complex mix of electro-mechanical and hydraulic components. A faulty component will manifest itself indirectly via a change in the characteristics of observed sensor readings in the pointing control loop. Because of the non-linearity and feedback present, direct causal relationships between fault conditions and observed symptoms can be difficult to establish — this makes manual fault diagnosis a slow and expensive process. In addition, if a pointing problem occurs while a spacecraft is being tracked, the antenna is often shut-down to prevent any potential damage to the structure, and the track is transferred to another antenna if possible. Hence, at present, diagnosis often occurs after the fact, where the original fault conditions may be difficult to replicate. An obvious strategy is to design an on-line automated monitoring system. Conventional control-theoretic models for fault detection are impractical due to the difficulties in constructing accurate models for such a non-linear system — an alternative is to learn the symptom-fault mapping directly from training data, the approach we follow here.

## 2    Fault Classification over Time

### 2.1    Data Collection and Feature Extraction

The observable data consists of various sensor readings (in the form of sampled time series) which can be monitored while the antenna is in tracking mode. The approach we take is to estimate the state of the system at discrete intervals in time. A feature vector $\underline{x}$ of dimension $k$ is estimated from sets of successive windows of sensor data. A pattern recognition component then models the instantaneous estimate of the posterior class probability given the features, $p(\omega_i|\underline{x})$, $1 \leq i \leq m$. Finally, a hidden Markov model is used to take advantage of temporal context and estimate class probabilities conditioned on recent past history. This hierarchical pattern of information flow, where the time series data is transformed and mapped into a categorical representation (the fault classes) and integrated over time to enable robust decision-making, is quite generic to systems which must passively sense and monitor their environment in real-time.

Experimental data was gathered from a new antenna at a research ground-station at the Goldstone DSN complex in California. We introduced hardware faults in a

controlled manner by switching faulty components in and out of the control loop. Obtaining data in this manner is an expensive and time-consuming procedure since the antenna is not currently instrumented for sensor data acquisition and is located in a remote location of the Mojave Desert in Southern California. Sensor variables monitored included wind speed, motor currents, tachometer voltages, estimated antenna position, and so forth, under three separate fault conditions (plus normal conditions).

The time series data was segmented into windows of 4 seconds duration (200 samples) to allow reasonably accurate estimates of the various features. The features consisted of order statistics (such as the range) and moments (such as the variance) of particular sensor channels. In addition we also applied an autoregressive-exogenous (ARX) modelling technique to the motor current data, where the ARX coefficients are estimated on each individual 4-second window of data. The autoregressive representation is particularly useful for discriminative purposes (Eggers and Khuon, 1990).

## 2.2   State Estimation with a Hidden Markov Model

If one applies a simple feed-forward network model to estimate the class probabilities at each discrete time instant $t$, the fact that faults are typically correlated over time is ignored. Rather than modelling the temporal dependence of features, $p(\underline{x}(t)|\underline{x}(t-1),\ldots,\underline{x}(0))$, a simpler approach is to model temporal dependence via the class variable using a Hidden Markov Model (HMM). The $m$ classes comprise the Markov model states. Components of the Markov transition matrix $\mathbf{A}$ (of dimension $m \times m$) are specified *subjectively* rather than estimated from the data, since there is no reliable database of fault-transition information available at the component level from which to estimate these numbers. The *hidden* component of the HMM model arises from the fact that one cannot observe the states directly, but only indirectly via a stochastic mapping from states to symptoms (the features). For the results reported in this paper, the state probability estimates at time $t$ are calculated using all the information available up to that point in time. The probability state vector is denoted by $p(\mathbf{s}(t))$. The probability estimate of state $i$ at time $t$ can be calculated recursively via the standard HMM equations:

$$\hat{\mathbf{u}}(t) = \mathbf{A}.p(\mathbf{s}(t-1)) \quad \text{and} \quad p(s_i(t)) = \frac{\hat{u}_i(t)y_i(t)}{\sum_{j=1}^{m} \hat{u}_j(t)y_j(t)}$$

where the estimates are initialised by a prior probability vector $p(\mathbf{s}(0))$, the $u_i(t)$ are the components of $\mathbf{u}(t)$, $1 \leq i \leq m$, and the $y_i(t)$ are the likelihoods $p(\underline{x}|\omega_i)$ produced by the particular classifier being used (which can be estimated to within a normalising constant by $p(\omega_i|\underline{x})/p(\omega_i)$).

## 2.3   Classification Results

We compare a feedforward multi-layer perceptron model (single hidden layer with 12 sigmoidal units, trained using the squared error objective function and a conjugate-gradient version of backpropagation) and a simple maximum-likelihood Gaussian classifier (with an assumed diagonal covariance matrix, variances estimated from the data), both with and without the HMM component. Table 1 summarizes the

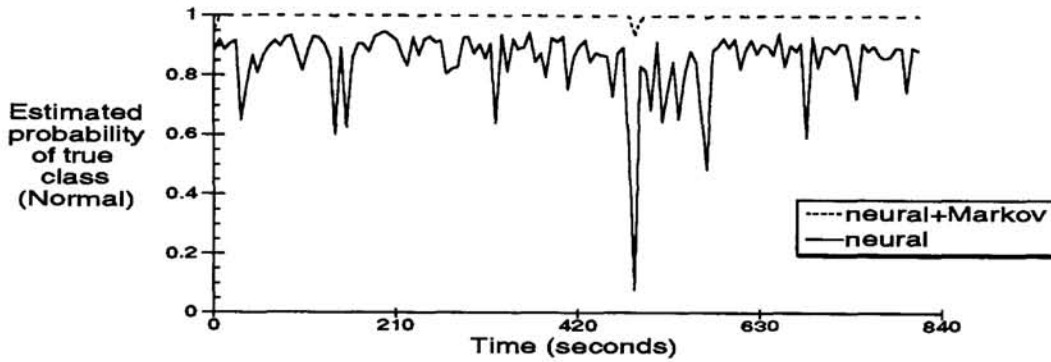

Figure 1: Stabilising effect of Markov component

overall classification accuracies obtained for each of the models — these results are for models trained on data collected in early 1991 (450 windows) which were then field-tested in real-time at the antenna site in November 1991 (596 windows). There were 12 features used in this particular experiment, including both ARX and time-domain features. Clearly, the neural-Markov model is the best model in the sense that no samples at all were misclassified — it is significantly better than the simple Gaussian classifier. Without the Markov component, the neural model still classified quite well (0.84% error rate). However all of its errors were false alarms (the classifier decision was a fault label, when in reality conditions were normal) which are highly undesirable from an operational viewpoint — in this context, the Markov model has significant practical benefit. Figure 1 demonstrates the stabilising effect of the Markov model over time. The vertical axis corresponds to the probability estimate of the model for the true class. Note the large fluctuations and general uncertainty in the neural output (due to the inherent noise in the feature data) compared to the stability when temporal context is modelled.

**Table 1: Classification results for different models**

|  | Percentage error rate in Field Test | |
|---|---|---|
| Model | Without HMM | With HMM |
| Gaussian model | 16.94 | 14.42 |
| Feedforward network | 0.84 | 0.0 |

## 3  Detecting novel classes

While the neural model described above exhibits excellent performance in terms of discrimination, there is another aspect to classifier performance which we must consider for applications of this nature: how will the classifier respond if presented with data from a class which was not included in the training set ? Ideally, one would like the model to detect this situation. For fault diagnosis the chance that one will encounter such novel classes under operational conditions is quite high since there is little hope of having an exhaustive library of faults to train on.

In general, whether one uses a neural network, decision tree or other classification

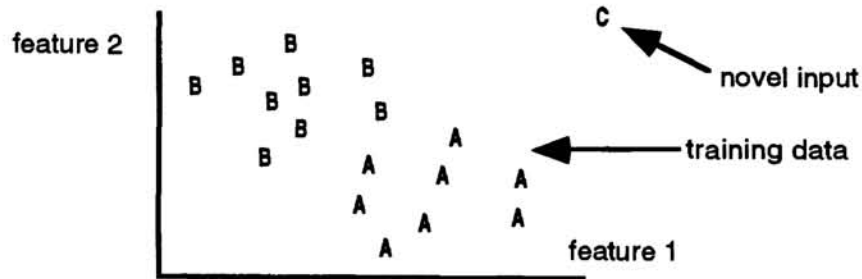

Figure 2: Data from a novel class $C$

method, there are few guarantees about the *extrapolation* behaviour of the trained classification model. Consider Figure 2, where point C is far away from the "A"s and "B"s on which the model is trained. The response of the trained model to point C may be somewhat arbitrary, since it may lie on either side of a decision boundary depending on a variety of factors such as initial conditions for the training algorithm, objective function used, particular training data, and so forth. One might hope that for a feedforward multi-layer perceptron, novel input vectors would lead to low response for all outputs. However, if units with non-local response functions are used in the model (such as the commonly used sigmoid function), the tendency of training algorithms such as backpropagation is to generate mappings which have a large response for at least one of the classes as the attributes take on values which extend well beyond the range of the training data values. Leonard and Kramer (1990) discuss this particular problem of poor extrapolation in the context of fault diagnosis of a chemical plant. The underlying problem lies in the basic nature of *discriminative* models which focus on estimating decision boundaries based on the differences between classes. In contrast, if one wants to detect data from novel classes, one must have a *generative* model for each known class, namely one which specifies how the data is generated for these classes. Hence, in a probabilistic framework, one seeks estimates of the probability density function of the data given a particular class, $f(\underline{x}|\omega_i)$, from which one can in turn use Bayes' rule for prediction:

$$p(\omega_i|\underline{x}) = \frac{f(\underline{x}|\omega_i)p(\omega_i)}{\sum_{j=1}^{m} f(\underline{x}|\omega_j)p(\omega_j)} \qquad (1)$$

## 4   Kernel Density Estimation

Unless one assumes a particular parametric form for $f(\underline{x}|\omega_i)$, then it must be somehow estimated from the data. Let us ignore the multi-class nature of the problem temporarily and simply look at a single-class case. We focus here on the use of *kernel*-based methods (Silverman, 1986). Consider the 1-dimensional case of estimating the density $f(x)$ given samples $\{x_i\}$, $1 \le i \le N$. The idea is simple enough: we obtain an estimate $\hat{f}(x)$, where $x$ is the point at which we wish to know the density, by summing the contributions of the kernel $K((x - x_i)/h)$ (where $h$ is the *bandwidth* of the estimator, and $K(.)$ is the *kernel function*) over all the samples

and normalizing such that the estimate is itself a density, i.e.,

$$\hat{f}(x) = \frac{1}{Nh} \sum_{i=1}^{N} K\left(\frac{x - x_i}{h}\right) \qquad (2)$$

The estimate $\hat{f}(x)$ directly inherits the properties of $K(.)$, hence it is common to choose the kernel shape itself to be some well-known smooth function, such as a Gaussian. For the multi-dimensional case, the product kernel is commonly used:

$$\hat{f}(\underline{x}) = \frac{1}{Nh_1...h_d} \sum_{i=1}^{N} \left(\prod_{k=1}^{d} K\left(\frac{x^k - x_i^k}{h_k}\right)\right) \qquad (3)$$

where $x^k$ denotes the component in dimension $k$ of vector $\underline{x}$, and the $h_i$ represent different bandwidths in each dimension.

Various studies have shown that the quality of the estimate is typically much more sensitive to the choice of the bandwidth $h$ than it is to the kernel shape $K(.)$ (Izenmann, 1991). Cross-validation techniques are usually the best method to estimate the bandwidths from the data, although this can be computationally intensive and the resulting estimates can have a high variance across particular data sets. A significant disadvantage of kernel models is the fact that all training data points must be stored and a distance measure between a new point and each of the stored points must be calculated for each class prediction. Another less obvious disadvantage is the lack of empirical results and experience with using these models for real-world applications — in particular there is a dearth of results for high-dimensional problems. In this context we now outline a *kernel approximation* model which is considerably simpler both to train and implement than the full kernel model.

## 5   Kernel Approximation using Mixture Densities

### 5.1   Generating a kernel approximation

An obvious simplification to the full kernel model is to replace clusters of data points by representative centroids, to be referred to as the *centroid kernel* model. Intuitively, the sum of the responses from a number of kernels is approximated by a single kernel of appropriate width. Omohundro (1992) has proposed algorithms for bottom-up merging of data points for problems of this nature. Here, however, we describe a top-down approach by observing that the kernel estimate is itself a special case of a *mixture* density. The underlying density is assumed to be a linear combination of $L$ mixture components, i.e.,

$$f(x) = \sum_{i=1}^{L} \alpha_i f_i(x) \qquad (4)$$

where the $\alpha_i$ are the mixing proportions. The full kernel estimate is itself a special case of a mixture model with $\alpha_i = 1/N$ and $f_i(x) = K(x)$. Hence, our centroid kernel model can also be treated as a mixture model but now the parameters of the mixture model (the mixing proportions or weights, and the widths and locations of the centroid kernels) must be estimated from the data. There is a well-known and

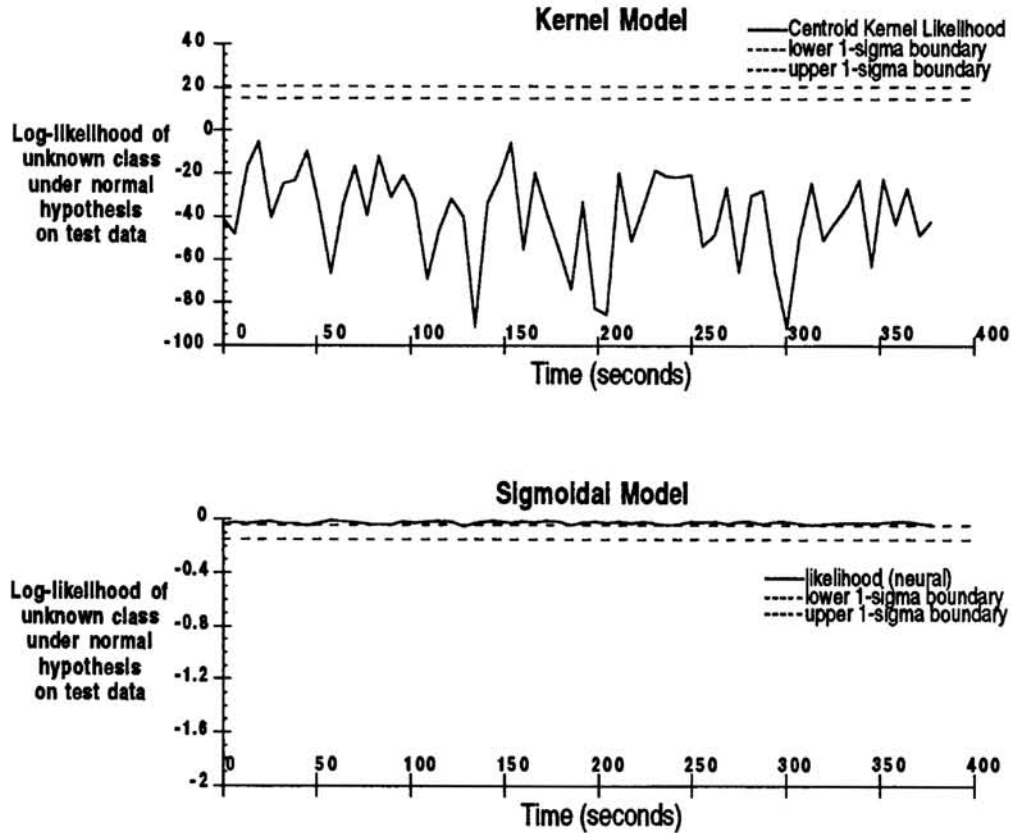

Figure 3: Likelihoods of kernel versus sigmoidal model on novel data

fast statistical procedure known as the EM (Expectation-Maximisation) algorithm for iteratively calculating these parameters, given some initial estimates (e.g., Redner and Walker, 1984). Hence, the procedure for generating a centroid kernel model is straightforward: divide the training data into homogeneous subsets according to class labels and then fit a mixture model with $L$ components to each class using the EM procedure (initialisation can be based on randomly selected prototypes). Prediction of class labels then follows directly from Bayes' rule (Equation (1)). Note that there is a strong similarity between mixture/kernel models and Radial Basis Function (RBF) networks. However, unlike the RBF models, we do not train the output layer of our network in order to improve discriminative performance as this would potentially destroy the desired probability estimation properties of the model.

## 5.2   Experimental results on detecting novel classes

In Figure 3 we plot the log-likelihoods, $\log f(\underline{x}|\omega_i)$, as a function of time, for both a centroid kernel model (Gaussian kernel, $L = 5$) and the single-hidden layer sigmoidal network described in Section 2.2. Both of these models have been trained on only 3 of the original 4 classes (the discriminative performance of the models was roughly equivalent), excluding one of the known classes. The inputs $\{\underline{x}_i\}$ to the models are data from this fourth class. The dashed lines indicate the $\mu \pm \sigma$ boundaries on the

log-likelihood for the normal class as calculated on the training data — this tells us the *typical* response of each model for class "normal" (note that the absolute values are irrelevant since the likelihoods have not been normalised via Bayes rule). For both models, the maximum response for the novel data came from the normal class. For the sigmoidal model, the novel response was actually greater than that on the training data — the network is very confident in its erroneous decision that the novel data belongs to class normal. Hence, in practice, the presence of a novel class would be completely masked. On the other hand, for the kernel model, the measured response on the novel data is significantly lower than that obtained on the training data. The classifier can directly calculate that it is highly unlikely that this new data belongs to any of the 3 classes on which the model was trained. In practice, for a centroid kernel model, the training data will almost certainly fit the model better than a new set of test data, even data from the same class. Hence, it is a matter of calibration to determine appropriate levels at which new data is deemed sufficiently unlikely to come from any of the known classes. Nonetheless, the main point is that a local kernel representation facilitates such detection, in contrast to models with global response functions (such as sigmoids).

In general, one does not expect a generative model which is not trained discriminatively to be fully competitive in terms of classification performance with discriminative models — on-going research involves developing hybrid discriminative-generative classifiers. In addition, on-line learning of novel classes once they have been detected is an interesting and important problem for applications of this nature. An initial version of the system we have described in this paper is currently undergoing test and evaluation for implementation at DSN antenna sites.

## Acknowledgements

The research described in this paper was performed at the Jet Propulsion Laboratory, California Institute of Technology, under a contract with the National Aeronautics and Space Administration and was supported in part by DARPA under grant number AFOSR-90-0199.

## References

M. Eggers and T. Khuon, 'Neural network data fusion concepts and application,' in *Proceedings of 1990 IJCNN* , San Diego, vol.II, 7–16, 1990.

M. A. Kramer and J. A. Leonard, 'Diagnosis using backpropagation neural networks — analysis and criticism,' *Computers in Chemical Engineering*, vol.14, no.12, pp.1323–1338, 1990.

B. Silverman, *Density Estimation for Statistics and Data Analysis*, New York: Chapman and Hall, 1986.

A. J. Izenmann, 'Recent developments in nonparametric density estimation,' *J. Amer. Stat. Assoc.*, vol.86, pp.205–224, March 1991.

S. Omohundro, 'Model-merging for improved generalization,' in this volume.

R. A. Redner and H. F. Walker, 'Mixture densities, maximum likelihood, and the EM algorithm,' *SIAM Review*, vol.26, no.2, pp.195–239, April 1984.
